# Analysis of Information in Speech based on MANOVA

**Sachin S. Kajarekar[1] and Hynek Hermansky[1,2]**
[1]Department of Electrical and Computer Engineering
OGI School of Science and Engineering at OHSU
Beaverton, OR
[2]International Computer Science Institute
Berkeley, CA
*{sachin,hynek}@asp.ogi.edu*

## Abstract

We propose analysis of information in speech using three sources - language (phone), speaker and channel. Information in speech is measured as mutual information between the source and the set of features extracted from speech signal. We assume that distribution of features can be modeled using Gaussian distribution. The mutual information is computed using the results of analysis of variability in speech. We observe similarity in the results of phone variability and phone information, and show that the results of the proposed analysis have more meaningful interpretations than the analysis of variability.

## 1 Introduction

Speech signal carries information about the linguistic message, the speaker, the communication channel. In the previous work [1, 2], we proposed analysis of information in speech as analysis of variability in a set of features extracted from the speech signal. The variability was measured as covariance of the features, and analysis was performed using using multivariate analysis of variance (MANOVA). Total variability was divided into three types of variabilities, namely, intra-phone (or phone) variability, speaker variability, and channel variability. Effect of each type was measured as its contribution to the total variability.

In this paper, we extend our previous work by proposing an information-theoretic analysis of information in speech. Similar to MANOVA, we assume that speech carries information from three main sources- language, speaker, and channel. We measure information from a source as mutual information (MI) [3] between the corresponding class labels and features. For example, linguistic information is measured as MI between phone labels and features. The effect of sources is measured in nats (or bits). In this work, we show it is easier to interpret the results of this analysis than the analysis of variability.

In general, MI between two random variables $X$ and $Y$ can be measured using three different methods [4]. First, assuming that $X$ and $Y$ have a joint Gaussian

distribution. However, we cannot use this method because one of the variables - a set of class labels - is discrete. Second, modeling distribution of $X$ or $Y$ using parametric form, for example, mixture of Gaussians [4]. Third, using non-parametric techniques to estimate distributions of $X$ and $Y$ [5]. The proposed analysis is based on the second method, where distribution of features is modeled as a Gaussian distribution. Although it is a strong assumption, we show that results of this analysis are similar to the results obtained using the third method [5].

The paper is organized as follows. Section 2 describes the experimental setup. Section 3 describes MANOVA and presents results of MANOVA. Section 4 proposes information theoretic approach for analysis of information in speech and presents the results. Section 5 compares these results with results from the previous study. Section 6 describes the summary and conclusions from this work.

## 2  Experimental Setup

In the previous work [1, 2], we have analyzed variability in the features using three databases - HTIMIT, OGI Stories and TIMIT. In this work, we present results of MANOVA using OGI Stories database; mainly for the comparison with Yang's results [5, 6]. English part of OGI Stories database consists of 207 speakers, speaking for approximately 1 minute each. Each utterance is transcribed at phone level. Therefore, phone is considered as a source of variability or source of information. The utterances are not labeled separately by speakers and channels, so we cannot measure speaker and channel as separate sources. Instead, we assume that different speakers have used different channels and consider speaker+channel as a single source of variability or a single source of information.

Figure 1 shows a commonly used time-frequency representation of energy in speech signal. The y-axis represents frequency, x-axis represents time, and the darkness of each element shows the energy at a given frequency and time. A spectral vector is defined by the number of points on the y-axis, $S(w, t_m)$. In this work, this vector contains 15 points on Bark spectrum. The vector is estimated at every 10 ms using a 25 ms speech segment. It is labeled by the phone and the speaker and channel label of the corresponding speech segment. A temporal vector is defined by a sequence of points along time at a given frequency, $S(w_n, t)$. In this work, it consists of 50 points each in the past and the future with respect to the current observation and the observation itself. As the spectral vectors are computed every 10 ms, the temporal vector represents 1 sec of temporal information. The temporal vectors are labeled by the phone and the speaker and channel label of the current speech segment. In this work, the analysis is performed independently using spectral and temporal vectors.

## 3  MANOVA

Multivariate analysis of variance (MANOVA) [7] is used to measure the variation in the data, $\{X \in R^n\}$, with respect to two or more factors. In this work, we use two factors - phone and speaker+channel. The underline model of MANOVA is

$$X_{ijk} = \bar{X}_{...} + \bar{X}_{i..} + \bar{X}_{ij.} + \epsilon_{ijk} \tag{1}$$

where, $i = 1, \cdots, p$, represents phones, $j = 1, \cdots sc$, represents speakers and channels. This equation shows that any feature vector, $X_{ijk}$, can be approximated using a sum of $\bar{X}_{..}$, the mean of the data; $\bar{X}_{i.}$, mean of the phone $i$; $\bar{X}_{ij.}$, mean of the speaker and channel $j$, and phone $i$; and $\epsilon_{ijk}$, an error in this approximation. Using

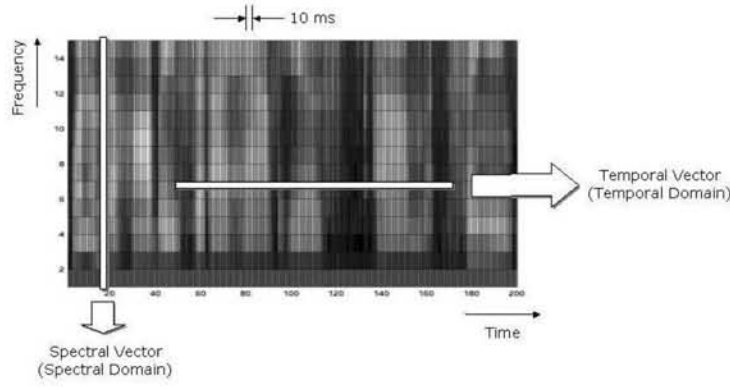

Figure 1: Time-frequency representation of logarithmic energies from speech signal

this model, the total covariance can be decomposed as follows

$$\Sigma_{total} = \Sigma_p + \Sigma_{sc} + \Sigma_{residual} \tag{2}$$

where

$$\Sigma_p = \sum_i \frac{N_i}{N} (\bar{X}_{i.} - \bar{X}_{..})^t (\bar{X}_{i.} - \bar{X}_{..})$$

$$\Sigma_{sc} = \sum_i \sum_j \frac{N_{ij}}{N} (\bar{X}_{ij} - \bar{X}_{i.})^t (\bar{X}_{ij} - \bar{X}_{i.})$$

$$\Sigma_{residual} = \frac{1}{N} \sum_i \sum_j \sum_k (X_{ijk} - \bar{X}_{ij})^t (X_{ijk} - \bar{X}_{ij})$$

and, $N$ is the data size and $N_{ijk}$ refers to the number of samples associated with the particular combination of factors (indicated by the subscript).

The covariance terms are computed as follows. First, all the feature vectors ($X$) belonging to each phone $i$ are collected and their mean ($\bar{X}_i$) is computed. The covariance of these phone means, $\Sigma_p$, is the estimate of phone variability. Next, the data for each speaker and channel $j$ within each phone $i$ is collected and the mean of the data ($\bar{X}_{ij}$) is computed. The covariance of the means of different speakers averaged over all phones, $\Sigma_{sc}$, is the estimate of speaker variability. All the variability in the data is not explained using these sources. The unaccounted sources, such as context and coarticulation, cause variability in the data collected from one speaker speaking one phone through one channel. The covariance within each phone, speaker, and channel is averaged over all phones, speakers, and channels, and the resulting covariance, $\Sigma_{residual}$, is the estimate of residual variability.

## 3.1 Results

Results of MANOVA are interpreted at two levels - feature element and feature vector. Results for each feature element are shown in Figure 2. Table 1 shows the results using the complete feature vector. The contribution of different sources is calculated as $trace(\Sigma_{source})/trace(\Sigma_{total})$. Note that this measure cannot be used to compare variabilities across feature-sets with different number of features. Therefore, we cannot directly compare contribution of variabilities in time and frequency domains. For comparison, contribution of sources in temporal domain is calculated

Table 1: Contribution of sources in spectral and temporal domains

| source | % contribution | |
| --- | --- | --- |
| | Spectral Domain | Temporal Domain |
| phone | 35.3 | 4.0 |
| speaker+channel | 41.1 | 30.3 |

as $trace(E^t\Sigma_{source}E)/trace(E^t\Sigma_{total}E)$, where $E_{101\times15}$ is a matrix of 15 leading eigenvectors of $\Sigma_{total}$.

In spectral domain, the highest phone variability is between 4-6 Barks. The highest speaker and channel variability is between 1-2 Barks where phone variability is the lowest. In temporal domain, phone variability spreads for approximately 250 ms around the current phone. Speaker and channel variability is almost constant except around the current frame. This deviation is explained by the difference in the phonetic context among the phone instances across different speakers. Thus, features for speakers within a phone differ not only because of different speaker characteristics but also different phonetic contexts. This deviation is also seen in the speaker and channel information in the proposed analysis. In the overall results for each domain, spectral domain has higher variability due to different phones than temporal domain. It also has higher speaker and channel variability than temporal domain.

The disadvantage of this analysis is that it is difficult to interpret the results. For example, how much phone variability is needed for perfect phone recognition? and is 4% of phone variability in temporal domain significant? In order to answer these questions, we propose an information theoretic analysis.

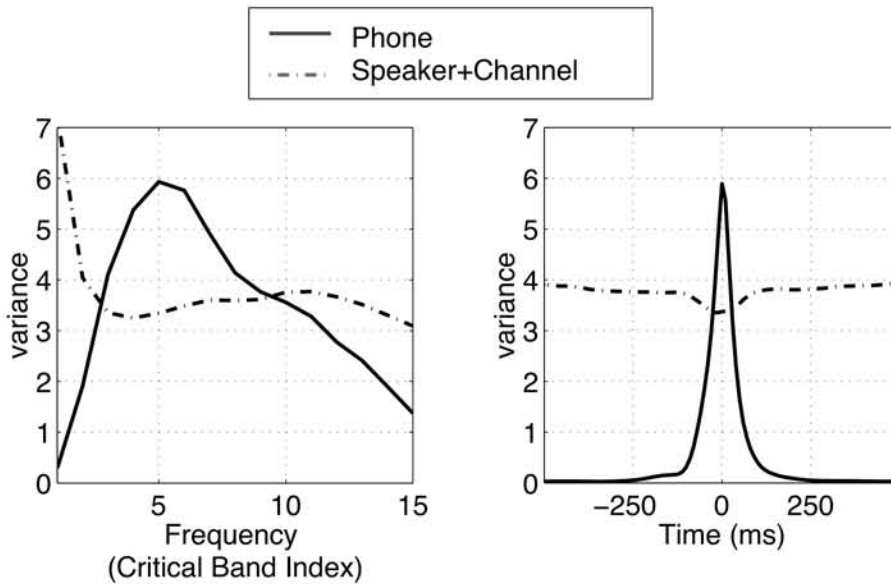

Figure 2: Results of analysis of variability

# 4 Information-theoretic Analysis

Results of MANOVA can not be directly converted to MI because the determinant of source and residual covariances do not add to the determinant of total covariance. Therefore, we propose a different formulation for the information theoretic analysis as follows. Let $\{X \in R^n\}$ be a set of feature vectors, with probability distribution $p(X)$. Let $h(X)$ be the entropy of $X$. Let $Y = \{Y_1, \ldots, Y_m\}$ be a set of different factors and each $Y_i$ be a set of classes within each factor. For example, we can assume that $Y_1 = \{y_1^i\}$ represents phone factor and each $y_1^i$ represent a phone class. Lets assume that $X$ has two parts; one completely characterized by $Y$ and another part, $Z$, characterized by $N(X) \sim \mathcal{N}(0, I_{n \times n})$, where $I$ is the identity matrix. Let $I(X;Y)$ be the MI between $X$ and $Y$. Assuming that we consider all the possible factors for our analysis,

$$I(X;Y) = I(X;Y_1, \ldots, Y_m) = h(X) - h(X/Y_1, \ldots, Y_m) = h(X) - h(Z) = D(P||N),$$

where $D()$ is the kullback-liebler distance [3] between distributions $P$ and $N$. Using the chain-rule, the left hand side can be expanded as follows,

$$I(X;Y_1, \ldots, Y_n) = I(X;Y_1) + I(X;Y_2/Y_1) + \sum_{i=3}^{m} I(X;Y_i/Y_{i-1}, \ldots, Y_2, Y_1). \quad (3)$$

If we assume that there are only two factors $Y_1$ and $Y_2$ used for the analysis, then this equation is similar to the decomposition performed using MANOVA (Equation 2). The term on the left hand side is entropy of $X$ which is the total information in $X$ that can be explained using $Y$. This is similar to the left-hand side term in MANOVA that describes the total variability. On the right hand side, first term is similar to the phone variability, second term is similar to the speaker variability, and the last term which calculates the effect of unaccounted factors $(Y_3, \ldots, Y_m)$ is similar to the residual variability.

First and second terms on the right hand side of Equation 3 are computed as follows.

$$I(X;Y_1) = h(X) - h(X/Y_1) \quad (4)$$

$$I(X;Y_2/Y_1) = h(X/Y_1) - h(X/Y_1, Y_2). \quad (5)$$

$h()$ terms are estimated using parametric approximation to the total and conditional distribution It is assumed that the total distribution of features is a Gaussian distribution with covariance $\Sigma$. Therefore, $h(X) = \frac{1}{2} \log (2\pi e)^n |\Sigma|$. Similarly, we assume that the distribution of features of different phones $(i)$ is a Gaussian distribution with covariances $\Sigma_i$. Therefore,

$$h(X/Y_1) = \frac{1}{2} \sum_{y_1^i \subset Y_i} p\left(y_1^i\right) \log (2\pi e)^n |\Sigma_i| \quad (6)$$

Finally, we assume that the distribution of features of different phones spoken by different speakers is also a Gaussian distribution with covariances $\Sigma_{ij}$. Therefore,

$$h(X/Y_1, Y_2) = \frac{1}{2} \sum_{y_1^i \subset Y_1, y_2^j \subset Y_2} p\left(y_1^i, y_2^j\right) \log (2\pi e)^n |\Sigma_{ij}| \quad (7)$$

Substituting equations 6 and 7 in equations 4 and 5, we get

$$I(X;Y_1) = \frac{1}{2} \log \frac{|\Sigma|}{\prod_{y_1^i \subset Y_i} |\Sigma_i|^{p(y_1^i)}} \quad (8)$$

$$I(X;Y_2/Y_1) = \frac{1}{2} \log \frac{\prod_{y_1^i \subset Y_i} |\Sigma_i|^{p(y_1^i)}}{\prod_{y_1^i \subset Y_1, y_2^j \subset Y_2} |\Sigma_i|^{p(y_1^i, y_2^j)}} \quad (9)$$

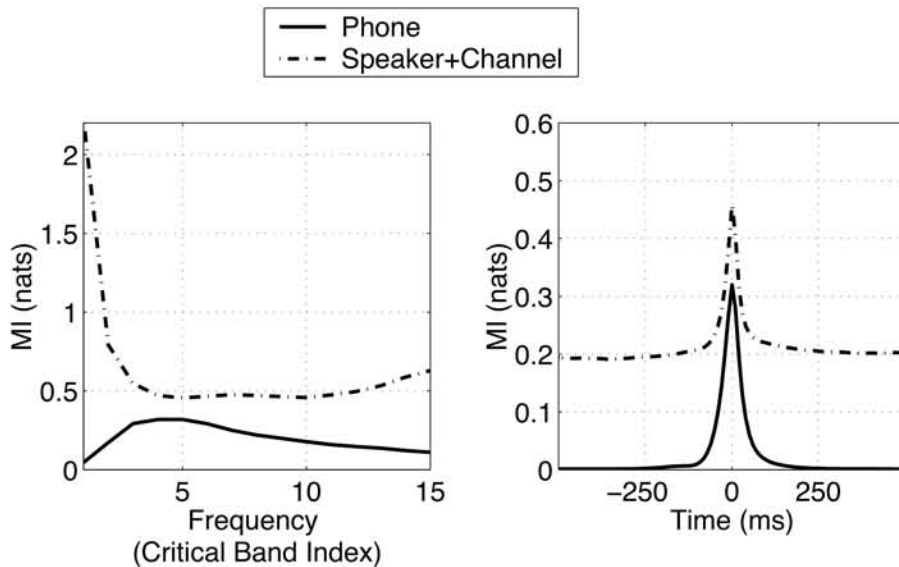

Figure 3: Results of information-theoretic analysis

Table 2: Mutual information between features and phone and speaker and channel labels in spectral and temporal domains

|  | MI (nats) | |
|---|---|---|
| source | Spectral Domain | Temporal Domain |
| phone | 1.6 | 1.2 |
| speaker+channel | 0.6 | 5.9 |

## 4.1 Results

Figure 3 shows the results of information-theoretic analysis in spectral and temporal domain. These results are computed independently for each feature element. In spectral domain, phone information is highest between 3-6 Barks. Speaker and channel information is lowest in that range and highest between 1-2 Barks. Since OGI Stories database was collected over different telephones, speaker+channel information below 2 Barks ( $\approx 200$ Hz ) is due to different telephone channels. In temporal domain, the highest phone information is at the center (0 ms). It spreads for approximately 200 ms around the center. Speaker and channel information is almost constant across time except near the center.

Note that the nature of speaker and channel variability also deviates from the constant around the current frame. But, at the current frame, phone variability is higher than speaker and channel variability. The results of analysis of information show that, at the current frame, phone information is lower than speaker and channel information. This difference is explained by comparing our MI results with results from Yang et. al. [6] in the next section.

Table 2 shows the results for the complete feature vector. Note that there are some practical issues in computing determinants in Equation 4 and 5. They are related to data insufficiency, specifically, in temporal domain where the feature vector is 101 points and there are approximately 60 vectors per speaker per phone. We ob-

serve that without proper conditioning of covariances, the analysis overestimates MI ($I(X; Y_1, Y_2) > H(Y_1, Y_2)$). This is addressed using the condition number to limit the number of eigenvalues used in the calculation of determinants. Our hypothesis is that in presence of insufficient data, only few leading eigen vectors are properly estimated. We have use condition number of 1000 to estimate determinant of $\Sigma$ and $\Sigma_i$, and condition number of 100 to estimate the determinant of $\Sigma_{ij}$. The results show that phone information in spectral domain is 1.6 nats. Speaker and channel information is 0.5 nats. In temporal domain, phone information is about 1.2 nats. Speaker and channel information is 5.9 nats. Comparison of results from spectral and temporal domains shows that spectral domain has higher phone information than temporal domain. Temporal domain has higher speaker and channel information than spectral domain.

Using these results, we can answer the questions raised in Section 3. First question was how much phone variability is needed for perfect phone recognition? The answer to the question is $H(Y_1)$, because the maximum value of $I(X; Y_1)$ is $H(Y_1)$. We compute $H(Y_1)$ using phone priors. For this database, we get $H(Y_1) = 3.42$ nats, that means we need 3.42 nats of information for perfect phone recognition. Question about significance of phone information in temporal domain is addressed by comparing it with information-less MI level. The information-less MI is computed as MI between the current phone label and features at 500 ms in the past or in the future. From our results, we get information-less MI equal to 0.0013 nats considering feature at 500 ms in the past, and 0.0010 nats considering features at 500 ms in the future[1]. The phone information in temporal domain is 1.2 bits that is greater than both the levels. Therefore it is significant.

## 5   Results in Perspective

In the proposed analysis, we estimated MI assuming Gaussian distribution for the features. This assumption is validated by comparing our results with the results from a study by Yang, et. al.,[6], where MI was computed without assuming any parametric model for the distribution of features. Note that only entropies can be directly compared for difference in the estimation technique [3]. However, MI using Gaussian assumption can be equal to, less or more than the actual MI. In the comparison of our results with Yang's results, we consider only the nature of information observed in both studies. The difference in actual MI levels across the two studies is related to the difference in the estimation techniques.

In spectral domain, Yang's study showed higher phone information between 3-8 Barks. The highest phone information was observed at 4 Barks. Higher speaker and channel information was observed around 1-2 Barks. In temporal domain, their study showed that phone information spreads for approximately 200 ms around the current time frame. Comparison of results from this analysis and our analysis shows that nature of phone information is similar in both studies. Nature of speaker and channel information in spectral domain is also similar. We could not compare the speaker and channel information in temporal domain because Yang's study did not present these results.

In Section 4.1, we observed difference in the nature of speaker and channel variability, and speaker and channel information at $f_i = 5$ Barks. Comparing MI levels from our study to those from Yang's study, we observe that Yang's results show that speaker and channel information at 5 Barks is less that the corresponding phone information. This is consistent with results of analysis of variability, but not with

the proposed analysis of information. As mentioned before, this difference is due to difference in the density estimation techniques used for computing MI. In the future work, we plan to model the densities using more sophisticated techniques, and improve the estimation of speaker and channel information.

## 6  Conclusions

We proposed analysis of information in speech using three sources of information - language (phone), speaker and channel. Information in speech was measured as MI between the class labels and the set of features extracted from speech signal. For example, linguistic information was measured using phone labels and the features. We modeled distribution of features using Gaussian distribution. Thus we related the analysis to previous proposed analysis of variability in speech. We observed similar results for phone variability and phone information. The speaker and channel variability and speaker and channel information around the current frame was different. This was shown to be related to the over-estimation of speaker and channel information using unimodal Gaussian model. Note that the analysis of information was proposed because its results have more meaningful interpretations than results of analysis of variability. For addressing the over-estimation, we plan to use more complex models ,such as mixture of Gaussians, for computing MI in the future work.

**Acknowledgments**

Authors thank Prof. Andrew Fraser from Portland State University for numerous discussions and helpful insights on this topic.

## Footnotes

[1]Information-less MI calculated using Yang et. al. is 0.019 bits

## References

[1] S. S. Kajarekar, N. Malayath and H. Hermansky, "Analysis of sources of variability in speech," in *Proc. of EUROSPEECH*, Budapest, Hungary, 1999.

[2] S. S. Kajarekar, N. Malayath and H. Hermansky, "Analysis of speaker and channel variability in speech," in *Proc. of ASRU*, Colorado, 1999.

[3] T. M. Cover and J. A. Thomas, *Elements of Information theory*, John Wiley & Sons, Inc., 1991.

[4] J. A. Bilmes, " Maximum Mutual Information Based Reduction Strategies for Cross-correlation Based Joint Distribution Modelling ," in *Proc. of ICASSP*, Seattle, USA, 1998.

[5] H. Hermansky H. Yang, S. van Vuuren, "Relevancy of Time-Frequency Features for Phonetic Classification Measured by Mutual Information," in *ICASSP'99*, Phoenix, Arizona, USA, 1999.

[6] H. H. Yang, S. Sharma, S. van Vuuren and H. Hermansky, "Relevance of Time–Frequency Features for Phonetic and Speaker-Channel Classification," *Speech Communication*, Aug. 2000.

[7] R. V. Hogg and E. A. Tannis, *Statistical Analysis and Inference*, PRANTICE HALL, fifth edition, 1997.
